# Inferring Interaction Networks using the IBP applied to microRNA Target Prediction

**Hai-Son Le**
Machine Learning Department
Carnegie Mellon University
Pittsburgh, PA, USA
hple@cs.cmu.edu

**Ziv Bar-Joseph**
Machine Learning Department
Carnegie Mellon University
Pittsburgh, PA, USA
zivbj@cs.cmu.edu

## Abstract

Determining interactions between entities and the overall organization and clustering of nodes in networks is a major challenge when analyzing biological and social network data. Here we extend the Indian Buffet Process (IBP), a nonparametric Bayesian model, to integrate noisy interaction scores with properties of individual entities for inferring interaction networks and clustering nodes within these networks. We present an application of this method to study how microRNAs regulate mRNAs in cells. Analysis of synthetic and real data indicates that the method improves upon prior methods, correctly recovers interactions and clusters, and provides accurate biological predictions.

## 1  Introduction

Determining interactions between entities based on observations is a major challenge when analyzing biological and social network data [1, 12, 15]. In most cases we can obtain information regarding each of the entities (individuals in social networks and proteins in biological networks) and some information about possible relationships between them (friendships or conversation data for social networks and motif or experimental data for biology). The goal is then to integrate these datasets to recover the interaction network between the entities being studied. To simplify the analysis of the data it is also beneficial to identify groups, or clusters, within these interaction networks. Such groups can then be mapped to specific demographics or interests in the case of social networks or to modules and pathways in biological networks [2].

A large number of generative models were developed to represent entities as members of a number of classes. Many of these models are based on the stochastic blockmodel introduced in [19]. While the number of classes in such models could be fixed, or provided by the user, nonparametric Bayesian methods have been applied to allow this number to be inferred based on the observed data [9]. The stochastic blockmodel was also further extended in [1] to allow mixed membership of entities within these classes. An alternate approach is to use latent features to describe entities. [10] proposed a nonparametric Bayesian matrix factorization method to learn the latent factors in relational data whereas [12] presented a nonparametric model to study binary link data. All of these methods rely on the pairwise link and interaction data and in most cases do not utilize properties of the individual entities when determining interactions.

Here we present a model that extends the Indian Buffet Process (IBP) [7], a nonparametric Bayesian prior over infinite binary matrices, to learn the interactions between entities with an unbounded number of groups. Specifically, we represent each group as a latent feature and define interactions between entities within each group. Such latent feature representation has been used in the past to describe entities [7, 10, 12] and IBP is an appropriate nonparametric prior to infer the number of latent features. However, unlike IBP our model utilizes interaction scores as priors and so the

model is not exchangeable anymore. We thus extend IBP by integrating it with Markov random field (MRF) constraints, specifically pairwise potentials as in Ising model. MRF priors has been combined with Dirichlet Process mixture models for image segmentation in a related work of Orbanz and Buhmann [13]. Pairwise information is also used in the distance dependent Chinese restaurant process [4] to encourage similar objects to be clustered. Our model is well suited for cases in which we are provided with information on both link structure and the outcome of the underlying interactions. In social networks such data can come from observations of conversation between individuals followed by actions of the specific individuals (for example, travel), whereas in biology it is suited for regulatory networks as discussed below.

We apply our model to study the microRNA (miRNA) target prediction problem. miRNAs were recently discovered as a class of regulatory RNA molecules that regulate the levels of messenger RNAs (mRNAs) (which are later translated to proteins) by binding and inhibiting their specific targets [15]. They were shown to play an important role in a number of diseases including cancer, and determining the set of genes that are targeted by each miRNA is an important question when studying these diseases. Several methods were proposed to predict targets of miRNAs based on their sequence[1]. While these predictions are useful, due to the short length of miRNAs, they lead to many false positives and some false negatives [8]. In addition to sequence information, it is now possible to obtain the expression levels of miRNAs and their predicted mRNA targets using microarrays. Since miRNAs inhibit their direct targets, integrating sequence and expression data can improve predictions regarding the interactions between miRNAs and their targets. A number of methods based on regression analysis were suggested for this task [8, 17]. While methods utilizing expression data improved upon methods that only used sequence data, they often treated each target mRNA in isolation. In contrast, it has now been shown that each miRNA often targets hundreds of genes, and that miRNAs often work in groups to achieve a larger impact [14]. Thus, rather than trying to infer a separate regression model for each mRNA we use our IBP extended model to infer a joint regression model for a cluster of mRNAs and the set of miRNAs that regulate them. Such a model would provide statistical confidence (since it combines several observations) while adhering more closely to the underlying biology. In addition to inferring the interactions in the dataset such a model would also provide a grouping for genes and miRNAs which can be used to improve function prediction.

## 2 Computational model

Firstly, we derive a distribution on infinite binary matrices starting with a finite model and taking the limit as the number of features goes to infinity. Secondly, we describe the application of our model to the miRNA target prediction problem using a Gaussian additive model.

### 2.1 Interaction model

Let $z_{ik}$ denote the $(i, k)$ entry of a matrix $\mathbf{Z}$ and let $z_k$ denote the $k$th column of $\mathbf{Z}$. The group membership of $N$ entities is defined by a (latent) binary matrix $\mathbf{Z}$ where $z_{ik} = 1$ if entity $i$ belongs to group $k$. Given $\mathbf{Z}$, we say that entity $i$ interacts with entity $j$ if $z_{ik}z_{jk} = 1$ for some $k$. Note that two entities can interact through many groups where each group represents one type of interaction. In many cases, a prior on such interactions can be obtained. Assume we have a $N \times N$ symmetric matrix $\mathbf{W}$, where $w_{ij}$ indicates the degree that we believe that entity $i$ and $j$ interact: $w_{ij} > 0$ if entities $i$ and $j$ are more likely to interact and $w_{ij} < 0$ if they are less likely to do so.

**Nonparametric prior for $\mathbf{Z}$**: Griffiths and Ghahramani [7] proposed the Indian Buffet Process (IBP) as a nonparametric prior distribution on sparse binary matrices $\mathbf{Z}$. The IBP can be derived from a simple stochastic process, described by a culinary metaphor. In this metaphor, there are $N$ customers (entities) entering a restaurant and choosing from an infinite array of dishes (groups). The first customer tries Poisson($\alpha$) dishes, where $\alpha$ is a parameter. The remaining customers enter one after the others. The $i$th customer tries a previously sampled dish $k$ with probability $\frac{m_k}{i}$, where $m_k$ is the number of previous customers who have sampled this dish. He then samples a Poisson($\frac{\alpha}{i}$) number of new dishes. This process defines an exchangeable distribution on the equivalence classes of $\mathbf{Z}$, which are the set of binary matrices that map to the same left-ordered binary matrices. [7]. Exchangeability

means that the order of the customers does not affect the distribution and that permutation of the data does not change the resulting likelihood.

The prior knowledge on interactions discussed above (encoded by $\mathbf{W}$) violates the exchangeability of the IBP since the group membership probability depends on the identities of the entities whereas exchangeability means that permutation of entities does not change the probability. In [11], Miller et al. presented the phylogenetic Indian Buffet Process (pIBP), where they used a tree representation to express non-exchangeability. In their model, the relationships among customers are encoded as a tree allowing them to exploit the sum-product algorithm in defining the updates for an MCMC sampler, without significantly increasing the computational burden when performing inference.

We combine the IBP with pairwise potentials using $\mathbf{W}$, constraining the dish selection of customers. Similar to the pIBP, the entries in $z_k$ are not chosen independently given $\pi_k$ but rather depend on the particular assignment of the remaining entries. In the following sections, we start with a model with a finite number of groups and consider the limit as the number of groups grows to derive the nonparametric prior. Note that in our model, as in the original IBP [7], while the number of rows are finite, the number of columns (features) could be infinite. We can thus define a prior on interactions between entities (since their number is known in advance) while still allowing for an infinite number of groups. This flexibility allows the group parameters to be drawn from an infinite mixtures of priors which may lead to identical groups of entities each with a different set of parameters.

### 2.1.1 Prior on finite matrices $\mathbf{Z}$

We have an $N \times K$ binary matrix $\mathbf{Z}$ where $N$ is the number of entities and $K$ is a fixed, finite number of groups. In the IBP, each group/column $k$ is associated with a parameter $\pi_k$, chosen from a $\text{Beta}(\alpha/K, 1)$ prior distribution where $\alpha$ is a hyperparameter:

$$\pi_k | \alpha \sim \text{Beta}\left(\frac{\alpha}{K}, 1\right) \qquad P(z_k | \pi_k) = \exp\left(\sum_i \left((1 - z_{ik})\log(1 - \pi_k) + z_{ik}\log\pi_k\right)\right)$$

The joint probability of a column $k$ and $\pi_k$ in the IBP is:

$$P(z_k, \pi_k | \alpha) = \frac{1}{B(\frac{\alpha}{K}, 1)} \exp\left(\sum_i \left((1 - z_{ik})\log(1 - \pi_k) + z_{ik}\log\pi_k\right) + \left(\frac{\alpha}{K} - 1\right)\log\pi_k\right) \quad (1)$$

where $B(\cdot)$ is the Beta function.

For our model, we add the new pairwise potentials on memberships of entities. Defining $\Phi_{z_k} = \exp\left(\sum_{i<j} w_{ij} z_{ik} z_{jk}\right)$, the joint probability of a column $k$ and $\pi_k$ is:

$$P(z_k, \pi_k | \alpha) = \frac{1}{Z'} \Phi_{z_k} \exp\left(\sum_i \left((1 - z_{ik})\log(1 - \pi_k) + z_{ik}\log\pi_k\right) + \left(\frac{\alpha}{K} - 1\right)\log\pi_k\right) \quad (2)$$

where $Z'$ is the partition function. Note that IBP is a special case of our model when all $w$'s are zeros ($\mathbf{W} = 0$).

Following [7], we define the lof-equivalence classes $[\mathbf{Z}]$ as the sets of binary matrices mapped to the same left-ordered binary matrices. The history $h_i$ of a feature k at an entity $i$ is defined as $(z_{1k}, \ldots, z_{(i-1)k})$. When no object is specified, $h$ refers to the full history. $m_k$ and $m_h$ denote the number of non-zero entries of a feature $k$ and a history $h$ respectively. $K_h$ is the number of features possessing the history $h$ while $K_0$ is the number of features having $m_k = 0$. $K_+ = \sum_{h=1}^{2^N - 1} K_h$ is the number of features for which $m_k > 0$.

By integrating over all values of $\pi_k$, we get the marginal probability of a binary matrix $\mathbf{Z}$.

$$P(\mathbf{Z}) = \prod_{k=1}^{K} \int_0^1 P(z_k, \pi_k | \alpha) \, d\pi_k \tag{3}$$

$$= \prod_{k=1}^{K} \frac{1}{Z'} \Phi_{z_k} \int_0^1 \exp\left(\left(\frac{\alpha}{K} + m_k - 1\right)\log\pi_k + (N - m_k)\log(1 - \pi_k)\right) d\pi_k \tag{4}$$

$$= \prod_{k=1}^{K} \frac{1}{Z'} \Phi_{z_k} B\left(\frac{\alpha}{K} + m_k, N - m_k + 1\right) \tag{5}$$

The partition function $Z'$ could be written as: $Z' = \sum_{h=0}^{2^N-1} \Phi_h B\left(\frac{\alpha}{K} + m_h, N - m_h + 1\right)$.

### 2.1.2 Taking the infinite limit

The probability of a particular lof-equivalence class of binary matrices, $[\mathbf{Z}]$, is:

$$P([\mathbf{Z}]) = \sum_{\mathbf{Z}} P(\mathbf{Z}) = \frac{K!}{\prod_{h=0}^{2^N-1} K_h!} \prod_{k=1}^{K} \frac{1}{Z'} \Phi_{z_k} B\left(m_k + \frac{\alpha}{K}, N - m_k + 1\right) \quad (6)$$

Taking the limit when $K \to \infty$, we can show that with $\Psi = \sum_{h=1}^{2^N-1} \Phi_h \frac{(N-m_h)!(m_h-1)!}{N!}$:

$$\lim_{K\to\infty} P([\mathbf{Z}]) = \lim_{K\to\infty} \frac{K!}{\prod_{h=0}^{2^N-1} K_h!} \prod_{k=1}^{K_+} \Phi_{z_k} \frac{B(m_k + \frac{\alpha}{K}, N - m_k + 1)}{B(\frac{\alpha}{K}, N + 1)} \prod_{k=1}^{K} \frac{1}{Z'} B(\frac{\alpha}{K}, N + 1) \quad (7)$$

$$= \frac{\alpha^{K+}}{\prod_{h=1}^{2^N-1} K_h!} \prod_{k=1}^{K_+} \Phi_{z_k} \frac{(N - m_k)!(m_k - 1)!}{N!} \exp\left(-\alpha\Psi\right) \quad (8)$$

The detailed derivations are shown in Appendix.

### 2.1.3 The generative process

We now describe a generative stochastic process for $\mathbf{Z}$. It can be understood by a culinary metaphor, where each row of $\mathbf{Z}$ corresponds to a customer and each column corresponds to a dish. We denote by $h(i)$ the value of $z_{ik}$ in the complete history $h$. With $\bar{\Phi}_h = \Phi_h \frac{(N-m_h)!(m_h-1)!}{N!}$, we define $\Psi_i = \sum_{h:h_i=0,h(i)=1} \bar{\Phi}_h$ so that $\Psi = \sum_{i=1}^{N} \Psi_i$. Finally, let $z_{<ik}$ be entries $1, \ldots, (i-1)$ of $z_k$.

Assume that we are provided with a compatibility score between pairs of customers. That is, we have a value $w_{ij}$ for the food preference similarity between customer $i$ and customer $j$. Higher values of $w_{ij}$ indicate similar preferences and customers with such values are more likely to select the same dish. Therefore, the dishes a customer selects may depend on the choices of previous customers. The first customer tries Poisson$(\alpha\Psi_1)$ dishes. The remaining customers enter one after the others. The $i$th customer selects dishes with a probability that partially depends on the selection of the previous customers. The probability that a dish would be selected is $\sum_{h:h_i=z_{<ik},h(i)=1} \bar{\Phi}_h / \sum_{h:h_i=z_{<ik}} \bar{\Phi}_h$. He then samples a Poisson$(\alpha\Psi_i)$ number of new dishes. This process repeats until all customers have made their selections. Although this process is not exchangeable, the sequential order of customers is not important. This means that we get the same marginal distribution for any particular order of customers. Let $K_1^{(i)}$ denote the number of new dishes sampled by customer $i$, the probability of a particular matrix generated by this process is:

$$P(\mathbf{Z}) = \frac{\alpha^{K+}}{\prod_{i=1}^{N} K_1^{(i)}} \prod_{k=1}^{K_+} \bar{\Phi}_{z_k} \exp\left(-\alpha\Psi\right) \quad (9)$$

If we only pay attention to the lof-equivalence classes $[\mathbf{Z}]$, since there are $\frac{\prod_{i=1}^{N} K_1^{(i)}}{\prod_{h=1}^{2^N-1} K_h!}$ matrices generated by this process mapped to the same equivalence classes, multiplying $P(\mathbf{Z})$ by this quantity recovers Equation (8). We show in Appendix that in the case of the IBP where $\Phi_h = 1$ for all histories $h$ (when $\mathbf{W} = 0$), this generative process simplifies to the Indian Buffet Process.

## 2.2 Regression model for mRNA expression

In this section, we describe the application using the nonparametric prior to the miRNA target prediction problem. However, the method is applicable in general settings where there is a way to model properties of one entity from properties of its interacting entities. Our input data are expression profiles of $M$ messenger RNA (mRNA) transcripts and $R$ miRNA transcript across $T$ samples. Let $\mathbf{X} = (x_1^T, \ldots, x_M^T)^T$, where each row vector $x_i$ is the expression profile of mRNA $i$ in all samples. Similarly, let $\mathbf{Y} = (y_1^T, \ldots, y_R^T)^T$ represent the expression profiles of $R$ miRNAs. Furthermore, suppose we are given a $M \times R$ matrix $\mathbf{C}$ where $c_{ij}$ is the prior likelihood score for the interaction of

mRNA $i$ and miRNA $j$. Such matrix $\mathbf{C}$ could be obtained from sequence-based miRNA target predictions as discussed above. Applying our interaction model to this problem, the set of $N = M + R$ entities are divided into two disjoint sets of mRNAs and miRNAs. Let $\mathbf{Z} = (\mathbf{U}^T, \mathbf{V}^T)^T$ where $\mathbf{U}$ and $\mathbf{V}$ are the group membership matrices for mRNAs and miRNAs respectively, $\mathbf{W}$ is given by $\begin{pmatrix} \mathbf{0} & \mathbf{C} \\ \mathbf{C}^T & \mathbf{0} \end{pmatrix}$. Therefore, mRNA $i$ and miRNA $j$ interact through all groups $k$ such that $u_{ik}v_{jk} = 1$.

### 2.2.1 Gaussian additive model

In the interaction model suggested by GenMiR++ [8], each miRNA expression profile is used to explain the downregulation of the expression of its targeted mRNAs. Our model uses a group specific and miRNA specific coefficients ( $\mathbf{s} = [s_1, \ldots, s_\infty]^T$, with $s_k > 0$ for groups and $\mathbf{r} = [r_1, \ldots, r_R]^T$ for all miRNAs) to model the downregulation effect. These coefficients represent the baseline effect of group members and the strength of specific miRNAs, respectively. Using these parameters the expression level of a specific mRNA could be explained by summing over expression profiles of all miRNAs targeting the mRNA:

$$x_i \sim \mathcal{N}\big(\mu - \sum_j (r_j + \sum_{k:u_{ik}v_{jk}=1} s_k)\, y_j,\ \sigma^2 I\big) \tag{10}$$

where $\mu$ represents baseline expression for this mRNA and $\sigma$ is used to represent measurement noise. Thus, under this model, the expression of a mRNA are reduced from their baseline values by a linear combination of expression values of the miRNAs that target them. The probability of the observed data given $\mathbf{Z}$ is: $P(\mathbf{X}, \mathbf{Y}|\mathbf{Z}, \Theta) \propto \exp\Big(-\frac{1}{2\sigma^2}\sum_i (x_i - \bar{x}_i)^T(x_i - \bar{x}_i)\Big)$, with $\Theta = \{\mu, \sigma^2, \mathbf{s}, \mathbf{r}\}$ and $\bar{x}_i = \mu - \sum_j (r_j + \sum_{k:u_{ik}v_{jk}=1} s_k)\, y_j$.

### 2.2.2 Priors for model variables

We use the following as prior distributions for the variables in our model:

$$s_k \sim \text{Gamma}(\alpha_s, \beta_s) \qquad\qquad \mathbf{r} \sim \mathcal{N}(0, \sigma_r^2 I) \tag{11}$$
$$\mu \sim \mathcal{N}(0, \sigma_\mu^2 I) \qquad\qquad 1/\sigma^2 \sim \text{Gamma}(\alpha_v, \beta_v)$$

where the $\alpha$ and $\beta$ are the shape and scale parameters. The parameters are given hyperpriors: $1/\sigma_r^2 \sim \text{Gamma}(a_r, b_r)$ and $1/\sigma_\mu^2 \sim \text{Gamma}(a_\mu, b_\mu)$. $\alpha_s, \beta_s, \alpha_v, \beta_v$ are also given Gamma hyperpriors.

## 3 Inference by MCMC

As with many nonparametric Bayesian models, exact inference is intractable. Instead we use a Markov Chain Monte Carlo (MCMC) method to sample from the posterior distribution of $\mathbf{Z}$ and $\Theta$. Although, our model allows $\mathbf{Z}$ to have infinite number of columns, we only need to keep track of non-zero columns of $\mathbf{Z}$, an important aspect which is exploited by several nonparametric Bayesian models [7]. Our sampling algorithm involves a mix of Gibbs and Metropolis-Hasting steps which are used to generate the new sample.

### 3.1 Sampling from populated columns of Z

Let $m_{-ik}$ is the number of one entries not including $z_{ik}$ in $z_k$. Also let $z_{-ik}$ denote the entries of $z_k$ except $z_{ik}$ and let $\mathbf{Z}_{-(ik)}$ be the entire matrix $\mathbf{Z}$ except $z_{ik}$. The probability of an entry given the remaining entries in a column can be derived by considering an ordering of customers such that customer $i$ is the last person in line and using the generative process in Section 2.1.3:

$$
\begin{aligned}
P(z_{ik} = 1 | z_{-ik}) &= \frac{\bar{\bar{\Phi}}_{z_{<ik}, z_{ik}=1}}{\bar{\bar{\Phi}}_{z_{<ik}, z_{ik}=1} + \bar{\bar{\Phi}}_{z_{<ik}, z_{ik}=0}} \\
&= \frac{\exp\big(\sum_{j\neq i} w_{ij} z_{jk}\big)(N - m_{-ik} - 1)!\, m_{-ik}!}{\exp\big(\sum_{j\neq i} w_{ij} z_{jk}\big)(N - m_{-ik} - 1)!\, m_{-ik}! + (N - m_{-ik})!(m_{-ik} - 1)!} \\
&= \frac{\exp\big(\sum_{j\neq i} w_{ij} z_{jk}\big) m_{-ik}}{\exp\big(\sum_{j\neq i} w_{ij} z_{jk}\big) m_{-ik} + (N - m_{-ik})}
\end{aligned}
$$

We could also get the result using the limiting probability in Equation (8). The probability of each $z_{ik}$ given all other variables is: $P(z_{ik}|\mathbf{X}, \mathbf{Y}, \mathbf{Z}_{-(ik)}) \propto P(\mathbf{X}, \mathbf{Y}|\mathbf{Z}_{-(ik)}, z_{ik})P(z_{ik}|z_{-ik})$. We need only to condition on $z_{-ik}$ since columns of $\mathbf{Z}$ are generated independently.

## 3.2 Sampling other variables

**Sampling a new column of Z:** New columns are columns that do not yet have any entries equal to 1 (empty groups). When sampling for an entity $i$, we assume this is the last customer in line. Therefore, based on the generative process described in Section 2.1.3, the number of new features are Poisson($\frac{\alpha}{N}$). For each new column, we need to sample a new group specific coefficient variable $s_k$. We can simply sample from the prior distribution given in Equation (11) since the probability $P(\mathbf{X}, \mathbf{Y}|\mathbf{Z}, \Theta)$ is not affected by these new columns since no interactions are currently represented by these columns.

**Sampling $s_k$ for populated columns:** Since we do not have a conjugate prior on $\mathbf{s}$, we cannot compute the conditional likelihood directly. We turn to Metropolis-Hasting to sample $\mathbf{s}$. The proposed distribution of a new value $s_k^*$ given the old value $s_k$ is $q(s_k^*|s_k) = \text{Gamma}(h, \frac{s_k}{h})$ where $h$ is the shape parameter. The mean of this distribution is the old value $s_k$. The acceptance ratio is

$$\mathcal{A}(s_k \to s_k^*) = \min\left[1, \frac{P(\mathbf{X}, \mathbf{Y}|\mathbf{Z}, \Theta \setminus \{s_k\}, s_k^*)\; p(s_k^*|\alpha_s, \beta_s)\; q(s_k|s_k^*)}{P(\mathbf{X}, \mathbf{Y}|\mathbf{Z}, \Theta)\; p(s_k|\alpha_s, \beta_s)\; q(s_k^*|s_k)}\right]$$

In our experiments, $h$ is selected so that the average acceptance rate is around 0.25 [5].

**Sampling $\mathbf{r}, \mu, \sigma^2$ and prior parameters:** Closed-form formulas for the posterior distributions of $\mathbf{r}, \mu$ and $\sigma^2$ can be derived due to their conjugacy. For example, the posterior distribution of $1/\sigma^2$ given the other variables is: $\text{Gamma}\left(\alpha_v + \frac{MT}{2}, \left(\frac{1}{\beta_v} + \frac{\sum_i (x_i - \bar{x}_i)^T (x_i - \bar{x}_i)}{2}\right)^{-1}\right)$. Equations for updates of $\mathbf{r}$ and $\mu$ are omitted due to lack of space. Gibbs sampling steps are used for $\sigma_r^2$ and $\sigma_\mu^2$ since we can compute the posterior distribution with conjugate priors. For prior parameters $\{\alpha_s, \beta_s, \alpha_v, \beta_v\}$, we use Metropolis-Hasting steps discussed previously.

## 4 Results

We name our method GroupMiR (Group MiRNA target prediction). In this section we compare the performance of GroupMiR with GenMiR++ [8], which is one of the popular methods for predicting miRNA-mRNA interactions. However, unlike our method it does not use grouping of mRNAs and attempts to predict each one separately. Besides, there are two other important differences of GenMiR++ from our method: *1*) GenMiR++ only consider interactions in the candidate set while our method consider all possible interactions. *2*) GenMiR++ accepts a binary matrix as a candidate set while our method allows continuous valued scores. To our best knowledge, GenMiR++, which uses the regression model for interaction between entities, is the only appropriate method[2] for comparison.

### 4.1 Synthetic data

We generated 9 synthetic datasets. Each dataset contains 20 miRNAs and 200 mRNAs. We set the number of groups to $K = 5$ and $T = 10$ for all datasets. The miRNA membership $\mathbf{V}$ is a random matrix with at most 5 ones in each column. The mRNA membership $\mathbf{U}$ is a random matrix with density of 0.1. The expression of mRNAs are generated from the model in Equation (10) with $\sigma^2 = 1$. The remaining random variables are sampled as follows: $y \sim \mathcal{N}(0, 1)$, $s \sim \mathcal{N}(1, 0.1)$ and $r \sim \mathcal{N}(0, 0.1)$. Since the sequence based predictions of miRNA-mRNA interactions are based on short complementary regions they often result in many more false positives than false negatives. We thus introduce noise to the true binary interaction matrix $\mathbf{C}'$ by probabilistically changing each zero value in that matrix to 1. We tested different noise probabilities: $0.1, 0.2, 0.4$ and $0.8$. We use $\mathbf{C} = 2\mathbf{C}' - 1.8$, $\alpha = 1$ and the hyperprior parameters are set to generic values. Our sampler is ran for 2000 iterations and 1000 iterations are discarded as burn-in.

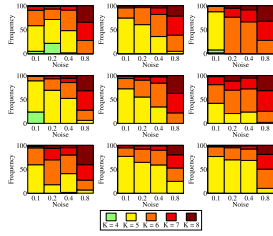

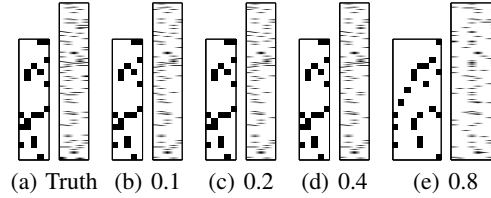

Figure 1: The posterior distribution of $K$.　　　Figure 2: An example synthetic dataset.

Figure 1 plots the estimated posterior distribution of $K$ from the samples of the 9 datasets for all noise levels. As can be seen, when the noise level is small (0.1), the distributions are correctly centered around $K = 5$. With increasing noise levels, the number of groups is overestimated. However, GroupMiR still does very well at a noise level of 0.4 and estimates for the higher noise level are also within a reasonable range.

We estimated a posterior mean for the interaction matrix **Z** by first ordering the columns of each sampled **Z** and then selecting the mode from the set of **Z** matrices. GenMiR++ returns a score value in $[0, 1]$ for each potential interaction. To convert these to binary interactions we tested a number of different threshold cutoffs: $0.5, 0.7$ and $0.9$. Figure 3 presents a number of quality measures for the recovered interactions by the two methods. GroupMiR achieves the best F1 score across all noise levels greatly improving upon GenMiR++ when high noise levels are considered (a reasonable biological scenario). In general, while the precision is very high for all noise levels, recall drops to a lower rate. From a biological point of view, precision is probably more important than recall since each of the predictions needs to be experimentally tested, a process that is often time consuming and expensive.

In addition to accurately recovering interactions between miRNAs and mRNAs, GroupMiR also correctly recovers the groupings of mRNA and miRNAs. Figure 2 presents a graphical view of the group membership in both the true model and the model recovered by GroupMiR for one of the synthetic datasets. As can be seen, our method is able to accurately recover the groupings of both miRNAs and mRNAs with moderate noise levels (up to 0.4). For the higher noise level (0.8) the method assigns more groups than in the underlying model. However, most interactions are still correctly recovered. These results hold for all datasets we tested (not shown due to lack of space).

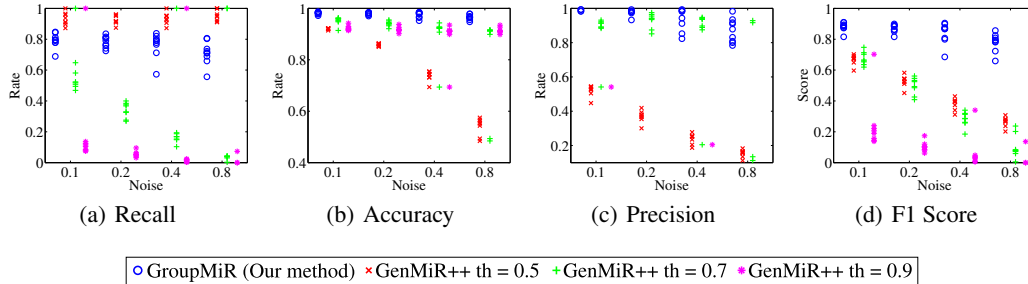

Figure 3: Performance of GroupMiR versus GenMiR++: Each data point is a synthetic dataset.

## 4.2 Application to mouse lung development

To test our method on real biological data, we used a mouse lung developmental dataset [6]. In this study, the authors used microarrays to profile both miRNAs and mRNAs at 7 time points, which include all recognized stages of lung development. We downloaded the log ratio normalized data collected in this study. Duplicate samples were averaged and median values of all probes were assigned to genes. As suggested in the paper, we used ratios to the last time point resulting in 6 values for each mRNA and miRNA. Priors for interaction between miRNA and mRNA were downloaded from the MicroCosm Target[3] database. Selecting genes with variance in the top $10\%$, led to 219 miRNAs and 1498 mRNAs which were used for further analysis.

We collected 5000 samples of the interaction matrix **Z** following a 5000 iteration burn-in period. Convergence of the MCMC chain is determined by monitoring trace plots of $K$ in multiple chains.

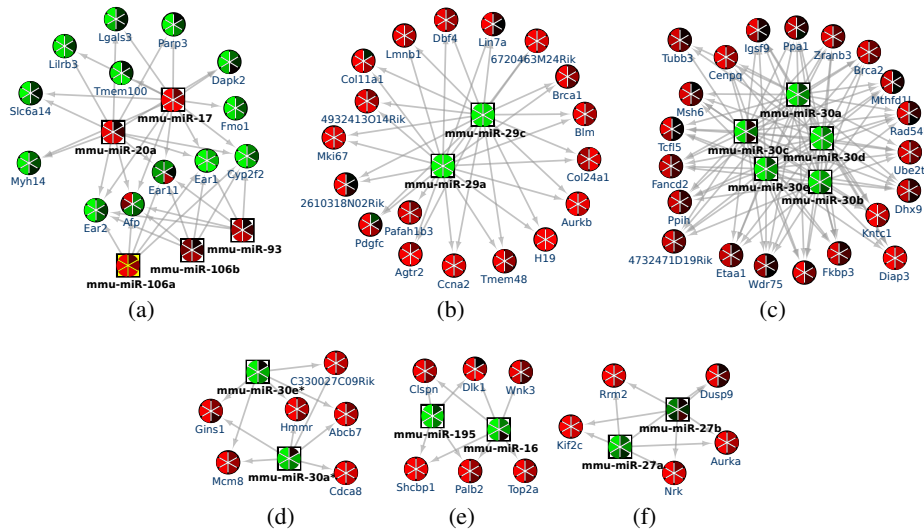

(a)        (b)        (c)

(d)        (e)        (f)

Figure 4: Interaction network recovered by GroupMiR: Each node is a pie chart corresponding to its expression values in the 6 time points (red: up-regulation, green: down-regulation).

Since there are many more entries for real data compared to synthetic data we computed a consensus for $\mathbf{Z}$ by reordering columns in each sample and averaging the entries across all matrices.

We further analyzed the network constructed from groups with at least $90\%$ posterior probability. The network recovered by GroupMiR is more connected (89 nodes and 208 edges) when compared to the network recovered by GenMiR++ (using equivalent 0.9 threshold) with 37 nodes and 34 edges (Appendix). We used Cytoscape [16] to visualize the 6 groups of interactions in Figure 4. The network contains several groups of co-expressed miRNAs controlling sets of mRNA, in agreement with previous biological studies [20].

To test the function of the clusters identified, we performed Gene Ontology (GO) enrichment analysis for the mRNAs using GOstat [3]. The full results (Bonferroni corrected) are presented in Appendix. As can be seen, several cell division categories are enriched in cluster (b) which is expected when dealing with a developing organ (which undergoes several rounds of cell division). Other significant functions include organelle organization and apoptosis which also are associated with development (cluster (c)). We performed similar GO enrichment analysis for the GenMiR++ results and for K-means when using the same set of mRNAs (setting $k = 6$ as in our model). In both cases we did not find any significant enrichment indicating that only by integrating sets of miRNAs with the mRNAs for this data we can find functional biological groupings. See Appendix for details.

We have also looked at the miRNAs controlling the different clusters and found that in a number of cases these agreed with prior knowledge. Cluster (a) includes 2 members of the miR 17-92 cluster, which is known to be critical to lung organogenesis [18]. MiRNA families miR-30, miR-29, miR-20 and miR-16, all identified by our method, were also reported to play roles in the early stages of lung organogenesis [6]. It is important to point out that we did not filter miRNAs explicitly based on expression but these miRNAs came in the results based on their strong effect on mRNA expression.

## 5 Conclusions

We have described an extension to IBP that allows us to integrate priors on interactions between entities with measured properties for individual entities when constructing interaction networks. The method was successfully applied to predict miRNA-mRNA interactions and we have shown that it works well on both synthetic and real data. While our focus in this paper was on a biological problem, several other datasets provide similar information including social networking data. Our method is appropriate for such datasets and can help when attempting to construct interaction networks based on observations.

### Acknowledgments

This work is supported in part by NIH grants 1RO1GM085022, 1U01HL108642 and NSF grant DBI-0965316 to Z.B.J.

## Footnotes

[1]Genes that are targets of miRNAs contain the reverse complement of part of the miRNA sequence.

[2]We also tested with the original IBP (by setting $\mathbf{W} = 0$). The results for both the synthetic and real data were too weak to be comparable with GenMIR++. See Appendix.

[3]http://www.ebi.ac.uk/enright-srv/microcosm/

# References

[1] E.M. Airoldi, D.M. Blei, S.E. Fienberg, and E.P. Xing. Mixed membership stochastic block-models. *The Journal of Machine Learning Research*, 9:1981–2014, 2008.

[2] Z. Bar-Joseph, G.K. Gerber, T.I. Lee, et al. Computational discovery of gene modules and regulatory networks. *Nature biotechnology*, 21(11):1337–1342, 2003.

[3] T. Beißbarth and T.P. Speed. GOstat: find statistically overrepresented Gene Ontologies within a group of genes. *Bioinformatics*, 20(9):1464, 2004.

[4] David M. Blei and Peter Frazier. Distance dependent chinese restaurant processes. In Johannes Frnkranz and Thorsten Joachims, editors, *ICML*, pages 87–94. Omnipress, 2010.

[5] S. Chib and E. Greenberg. Understanding the metropolis-hastings algorithm. *Amer. Statistician*, 49(4):327–335, 1995.

[6] J. Dong, G. Jiang, Y.W. Asmann, S. Tomaszek, et al. MicroRNA Networks in Mouse Lung Organogenesis. *PloS one*, 5(5):4645–4652, 2010.

[7] T. Griffiths and Z. Ghahramani. Infinite latent feature models and the Indian buffet process. *In Advances in Neural Information Processing Systems*, 18:475, 2006.

[8] J.C. Huang, T. Babak, T.W. Corson, et al. Using expression profiling data to identify human microRNA targets. *Nature methods*, 4(12):1045–1049, 2007.

[9] C. Kemp, J.B. Tenenbaum, T.L. Griffiths, , et al. Learning systems of concepts with an infinite relational model. In *Proc. 21st Natl Conf. Artif. Intell.(1)*, page 381, 2006.

[10] E. Meeds, Z. Ghahramani, R.M. Neal, and S.T. Roweis. Modeling dyadic data with binary latent factors. *In Advances in NIPS*, 19:977, 2007.

[11] K.T. Miller, T.L. Griffiths, and M.I. Jordan. The phylogenetic indian buffet process: A non-exchangeable nonparametric prior for latent features. In *UAI*, 2008.

[12] K.T. Miller, T.L. Griffiths, and M.I. Jordan. Nonparametric latent feature models for link prediction. *In Advances in Neural Information Processing Systems*, 2009.

[13] P. Orbanz and J.M. Buhmann. Nonparametric bayesian image segmentation. *International Journal of Computer Vision*, 77(1):25–45, 2008.

[14] ME Peter. Targeting of mrnas by multiple mirnas: the next step. *Oncogene*, 29(15):2161–2164, 2010.

[15] N. Rajewsky. microRNA target predictions in animals. *Nature genetics*, 38:S8–S13, 2006.

[16] P. Shannon, A. Markiel, O. Ozier, et al. Cytoscape: a software environment for integrated models of biomolecular interaction networks. *Genome research*, 13(11):2498, 2003.

[17] F. Stingo, Y. Chen, M. Vannucci, et al. A Bayesian graphical modeling approach to microRNA regulatory network inference. *Ann. Appl. Statist*, 2010.

[18] A. Ventura, A.G. Young, M.M. Winslow, et al. Targeted Deletion Reveals Essential and Overlapping Functions of the miR-17-92 Family of miRNA Clusters. *Cell*, 132:875–886, 2008.

[19] Y.J. Wang and G.Y. Wong. Stochastic blockmodels for directed graphs. *Journal of the American Statistical Association*, 82(397):8–19, 1987.

[20] C. Xiao and K. Rajewsky. MicroRNA control in the immune system: basic principles. *Cell*, 136(1):26–36, 2009.

